# The discriminant center-surround hypothesis for bottom-up saliency

**Dashan Gao**      **Vijay Mahadevan**      **Nuno Vasconcelos**
Department of Electrical and Computer Engineering
University of California, San Diego
{dgao, vmahadev, nuno}@ucsd.edu

## Abstract

The classical hypothesis, that bottom-up saliency is a center-surround process, is combined with a more recent hypothesis that all saliency decisions are optimal in a decision-theoretic sense. The combined hypothesis is denoted as discriminant center-surround saliency, and the corresponding optimal saliency architecture is derived. This architecture equates the saliency of each image location to the discriminant power of a set of features with respect to the classification problem that opposes stimuli at center and surround, at that location. It is shown that the resulting saliency detector makes accurate quantitative predictions for various aspects of the psychophysics of human saliency, including non-linear properties beyond the reach of previous saliency models. Furthermore, it is shown that discriminant center-surround saliency can be easily generalized to various stimulus modalities (such as color, orientation and motion), and provides optimal solutions for many other saliency problems of interest for computer vision. Optimal solutions, under this hypothesis, are derived for a number of the former (including static natural images, dense motion fields, and even dynamic textures), and applied to a number of the latter (the prediction of human eye fixations, motion-based saliency in the presence of ego-motion, and motion-based saliency in the presence of highly dynamic backgrounds). In result, discriminant saliency is shown to predict eye fixations better than previous models, and produces background subtraction algorithms that outperform the state-of-the-art in computer vision.

## 1   Introduction

The psychophysics of visual saliency and attention have been extensively studied during the last decades. As a result of these studies, it is now well known that saliency mechanisms exist for a number of classes of visual stimuli, including color, orientation, depth, and motion, among others. More recently, there has been an increasing effort to introduce computational models for saliency. One approach that has become quite popular, both in the biological and computer vision communities, is to equate saliency with center-surround differencing. It was initially proposed in [12], and has since been applied to saliency detection in both static imagery and motion analysis, as well as to computer vision problems such as robotics, or video compression. While difference-based modeling is successful at replicating many observations from psychophysics, it has three significant limitations. First, it does not explain those observations in terms of fundamental computational principles for neural organization. For example, it implies that visual perception relies on a linear measure of similarity (difference between feature responses in center and surround). This is at odds with well known properties of higher level human judgments of similarity, which tend not to be symmetric or even compliant with Euclidean geometry [20]. Second, the psychophysics of saliency offers strong evidence for the existence of both non-linearities and asymmetries which are not easily reconciled with this model. Third, although the center-surround hypothesis intrinsically poses

saliency as a classification problem (of distinguishing center from surround), there is little basis on which to justify difference-based measures as optimal in a classification sense. From an evolutionary perspective, this raises questions about the biological plausibility of the difference-based paradigm.

An alternative hypothesis is that all saliency decisions are *optimal in a decision-theoretic sense*. This hypothesis has been denoted as discriminant saliency in [6], where it was somewhat narrowly proposed as the justification for a top-down saliency algorithm. While this algorithm is of interest only for object recognition, the hypothesis of decision theoretic optimality is much more general, and applicable to any form of center-surround saliency. This has motivated us to test its ability to explain the psychophysics of human saliency, which is better documented for the bottom-up neural pathway. We start from the combined hypothesis that 1) bottom-up saliency is based on center-surround processing, and 2) this processing is optimal in a decision theoretic sense. In particular, it is hypothesized that, in the absence of high-level goals, the most salient locations of the visual field are those that enable the discrimination between center and surround with smallest expected probability of error. This is referred to as the *discriminant center-surround hypothesis* and, by definition, produces saliency measures that are optimal in a classification sense. It is also clearly tied to a larger principle for neural organization: that all perceptual mechanisms are optimal in a decision-theoretic sense.

In this work, we present the results of an experimental evaluation of the plausibility of the discriminant center-surround hypothesis. Our study evaluates the ability of saliency algorithms, that are optimal under this hypothesis, to both

- reproduce subject behavior in classical psychophysics experiments, and
- solve saliency problems of practical significance, with respect to a number of classes of visual stimuli.

We derive decision-theoretic optimal center-surround algorithms for a number of saliency problems, ranging from static spatial saliency, to motion-based saliency in the presence of egomotion or even complex dynamic backgrounds. Regarding the ability to replicate psychophysics, the results of this study show that discriminant saliency not only replicates all anecdotal observations that can be explained by linear models, such as that of [12], but can also make (surprisingly accurate) quantitative predictions for non-linear aspects of human saliency, which are beyond the reach of the existing approaches. With respect to practical saliency algorithms, they show that discriminant saliency not only is more accurate than difference-based methods in predicting human eye fixations, but actually produces background subtraction algorithms that outperform the state-of-the-art in computer vision. In particular, it is shown that, by simply modifying the probabilistic models employed in the (decision-theoretic optimal) saliency measure - from well known models of natural image statistics, to the statistics of simple optical-flow motion features, to more sophisticated dynamic texture models - it is possible to produce saliency detectors for either static or dynamic stimuli, which are insensitive to background image variability due to texture, egomotion, or scene dynamics.

## 2  Discriminant center-surround saliency

A common hypothesis for bottom-up saliency is that the saliency of each location is determined by how distinct the stimulus at the location is from the stimuli in its surround (e.g., [11]). This hypothesis is inspired by the ubiquity of "center-surround" mechanisms in the early stages of biological vision [10]. It can be combined with the hypothesis of decision-theoretic optimality, by defining a classification problem that equates

- the class of interest, at location $l$, with the observed responses of a pre-defined set of features $\mathbf{X}$ within a neighborhood $\mathcal{W}_l^1$ of $l$ (the *center*),
- the null hypothesis with the responses within a surrounding window $\mathcal{W}_l^0$ (the *surround* ),

The saliency of location $l^*$ is then equated with the power of the feature set $\mathbf{X}$ to discriminate between *center* and *surround*. Mathematically, the feature responses within the two windows are interpreted as observations drawn from a random process $\mathbf{X}(l) = (X_1(l), \ldots, X_d(l))$, of dimension $d$, conditioned on the state of a hidden random variable $Y(l)$. The observed feature vector at any location $j$ is denoted by $\mathbf{x}(j) = (x_1(j), \ldots, x_d(j))$, and feature vectors $\mathbf{x}(j)$ such that $j \in \mathcal{W}_l^c, c \in$

$\{0, 1\}$ are drawn from class $c$ (i.e., $Y(l) = c$), according to conditional densities $P_{\mathbf{X}(l)|Y(l)}(\mathbf{x}|c)$. The saliency of location $l$, $S(l)$, is quantified by the mutual information between features, $\mathbf{X}$, and class label, $Y$,

$$S(l) = I_l(\mathbf{X}; Y) = \sum_c \int p_{\mathbf{X}(l), Y(l)}(\mathbf{x}, c) \log \frac{p_{\mathbf{X}(l), Y(l)}(\mathbf{x}, c)}{p_{\mathbf{X}(l)}(\mathbf{x}) p_{Y(l)}(c)} d\mathbf{x}. \tag{1}$$

The $l$ subscript emphasizes the fact that the mutual information is defined locally, within $\mathcal{W}_l$. The function $S(l)$ is referred to as the *saliency map*.

## 3 Discriminant saliency detection in static imagery

Since human saliency has been most thoroughly studied in the domain of static stimuli, we first derive the optimal solution for discriminant saliency in this domain. We then study the ability of the discriminant center-surround saliency hypothesis to explain the fundamental properties of the psychophysics of pre-attentive vision.

### 3.1 Feature decomposition

The building blocks of the static discriminant saliency detector are shown in Figure 1. The first stage, feature decomposition, follows the proposal of [11], which closely mimics the earliest stages of biological visual processing. The image to process is first subject to a feature decomposition into an intensity map and four broadly-tuned color channels, $I = (r + g + b)/3$, $R = \lfloor \tilde{r} - (\tilde{g} + \tilde{b})/2 \rfloor_+$, $G = \lfloor \tilde{g} - (\tilde{r} + \tilde{b})/2 \rfloor_+$, $B = \lfloor \tilde{b} - (\tilde{r} + \tilde{g})/2 \rfloor_+$, and $Y = \lfloor (\tilde{r} + \tilde{g})/2 - |\tilde{r} - \tilde{g}|/2 \rfloor_+$, where $\tilde{r} = r/I, \tilde{g} = g/I, \tilde{b} = b/I$, and $\lfloor x \rfloor_+ = \max(x, 0)$. The four color channels are, in turn, combined into two color opponent channels, $R - G$ for red/green and $B - Y$ for blue/yellow opponency. These and the intensity map are convolved with three Mexican hat wavelet filters, centered at spatial frequencies $0.02$, $0.04$ and $0.08$ cycle/pixel, to generate nine feature channels. The feature space $\mathcal{X}$ consists of these channels, plus a Gabor decomposition of the intensity map, implemented with a dictionary of zero-mean Gabor filters at 3 spatial scales (centered at frequencies of $0.08$, $0.16$, and $0.32$ cycle/pixel) and 4 directions (evenly spread from 0 to $\pi$).

### 3.2 Leveraging natural image statistics

In general, the computation of (1) is impractical, since it requires density estimates on a potentially high-dimensional feature space. This complexity can, however, be drastically reduced by exploiting a well known statistical property of band-pass natural image features, e.g. Gabor or wavelet coefficients: that features of this type exhibit strongly *consistent* patterns of dependence (bow-tie shaped conditional distributions) across a very wide range of classes of natural imagery [2, 9, 21]. The consistency of these feature dependencies suggests that they are, in general, not greatly informative about the image class [21, 2] and, in the particular case of saliency, about whether the observed feature vectors originate in the center or surround. Hence, (1) can usually be well approximated by the sum of marginal mutual informations [21][1], i.e.,

$$S(l) = \sum_{i=1}^{d} I_l(X_i; Y). \tag{2}$$

Since (2) only requires estimates of marginal densities, it has significantly less complexity than (1). This complexity can, indeed, be further reduced by resorting to the well known fact that the marginal densities are accurately modeled by a generalized Gaussian distribution (GGD) [13]. In this case, all computations have a simple closed form [4] and can be mapped into a neural network that replicates the standard architecture of V1: a cascade of linear filtering, divisive normalization, quadratic non-linearity and spatial pooling [7].

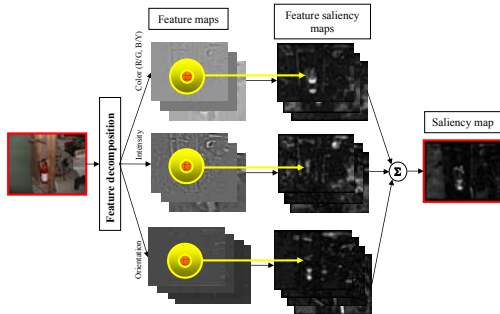

Figure 1: Bottom-up discriminant saliency detector.

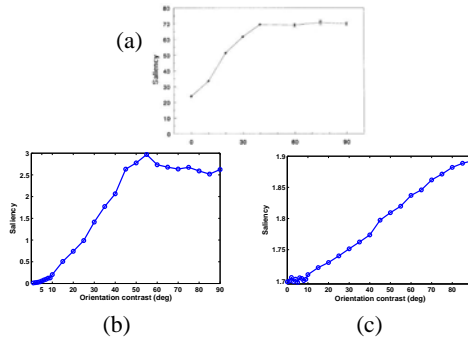

Figure 2: The nonlinearity of human saliency responses to orientation contrast [14] (a) is replicated by discriminant saliency (b), but not by the model of [11] (c).

## 3.3 Consistency with psychophysics

To evaluate the consistency of discriminant saliency with psychophysics, we start by applying the discriminant saliency detector to a series of displays used in classical studies of visual attention [18, 19, 14][2]. In [7], we have shown that discriminant saliency reproduces the anecdotal properties of saliency - percept of pop-out for single feature search, disregard of feature conjunctions, and search asymmetries for feature presence vs. absence - that have previously been shown possible to replicate with linear saliency models [11]. Here, we focus on *quantitative predictions* of human performance, and compare the output of discriminant saliency with both human data and that of the difference-based center-surround saliency model [11][3].

The first experiment tests the ability of the saliency models to predict a well known nonlinearity of human saliency. Nothdurft [14] has characterized the saliency of pop-out targets due to orientation contrast, by comparing the conspicuousness of orientation defined targets and luminance defined ones, and using luminance as a reference for relative target salience. He showed that the saliency of a target increases with orientation contrast, but in a non-linear manner: 1) there exists a threshold below which the effect of pop-out vanishes, and 2) above this threshold saliency increases with contrast, saturating after some point. The results of this experiment are illustrated in Figure 2, which presents plots of saliency strength vs orientation contrast for human subjects [14] (in (a)), for discriminant saliency (in (b)), and for the difference-based model of [11]. Note that discriminant saliency closely predicts the strong threshold and saturation effects characteristic of subject performance, but the difference-based model shows no such compliance.

The second experiment tests the ability of the models to make accurate quantitative predictions of search asymmetries. It replicates the experiment designed by Treisman [19] to show that the asymmetries of human saliency comply with Weber's law. Figure 3 (a) shows one example of the displays used in the experiment, where the central target (vertical bar) differs from distractors (a set of identical vertical bars) only in length. Figure 3 (b) shows a scatter plot of the values of discriminant saliency obtained across the set of displays. Each point corresponds to the saliency at the target location in one display, and the dashed line shows that, like human perception, discriminant saliency follows Weber's law: target saliency is approximately linear in the ratio between the difference of target/distractor length ($\Delta x$) and distractor length ($x$). For comparison, Figure 3 (c) presents the corresponding scatter plot for the model of [11], which clearly does not replicate human performance.

## 4 Applications of discriminant saliency

We have, so far, presented quantitative evidence in support of the hypothesis that pre-attentive vision implements decision-theoretical center-surround saliency. This evidence is strengthened by the

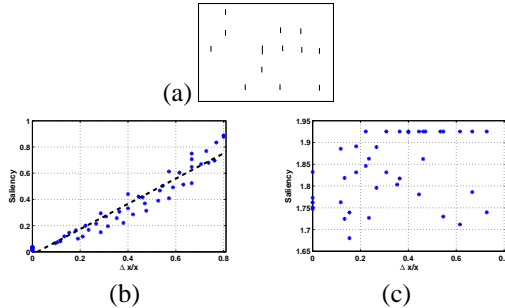

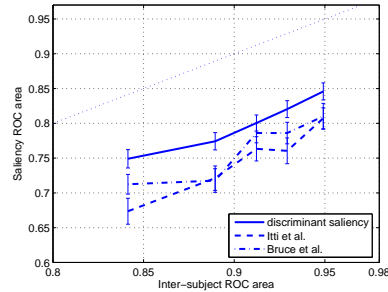

Figure 3: An example display (a) and performance of saliency detectors (discriminant saliency (b) and [11] (c)) on Weber's law experiment.

Figure 4: Average ROC area, as a function of inter-subject ROC area, for the saliency algorithms.

| Saliency model | Discriminant | Itti et al. [11] | Bruce et al. [1] |
|---|---|---|---|
| ROC area | 0.7694 | 0.7287 | 0.7547 |

Table 1: ROC areas for different saliency models with respect to all human fixations.

already mentioned one-to-one mapping between the discriminant saliency detector proposed above and the standard model for the neurophysiology of V1 [7]. Another interesting property of discriminant saliency is that its optimality is independent of the stimulus dimension under consideration, or of specific feature sets. In fact, (1) can be applied to any type of stimuli, and any type of features, as long as it is possible to estimate the required probability distributions from the center and surround neighborhoods. This encouraged us to derive discriminant saliency detectors for various computer vision applications, ranging from the prediction of human eye fixations, to the detection of salient moving objects, to background subtraction in the context of highly dynamic scenes. The outputs of these discriminant saliency detectors are next compared with either human performance, or the state-of-the-art in computer vision for each application.

## 4.1   Prediction of eye fixations on natural images

We start by using the static discriminant saliency detector of the previous section to predict human eye fixations. For this, the saliency maps were compared to the eye fixations of human subjects in an image viewing task. The experimental protocol was that of [1], using fixation data collected from 20 subjects and 120 natural images. Under this protocol, all saliency maps are first quantized into a binary mask that classifies each image location as either a fixation or non-fixation [17]. Using the measured human fixations as ground truth, a receiver operator characteristic (ROC) curve is then generated by varying the quantization threshold. Perfect prediction corresponds to an ROC area (area under the ROC curve) of 1, while chance performance occurs at an area of 0.5. The predictions of discriminant saliency are compared to those of the methods of [11] and [1].

Table 1 presents average ROC areas for all detectors, across the entire image set. It is clear that discriminant saliency achieves the best performance among the three detectors. For a more detailed analysis, we also plot (in Figure 4) the ROC areas of the three detectors as a function of the "inter-subject" ROC area (a measure of the consistency of eye movements among human subjects [8]), for the first two fixations - which are more likely to be driven by bottom-up mechanisms than the later ones [17]. Again, discriminant saliency exhibits the strongest correlation with human performance, this happens at all levels of inter-subject consistency, and the difference is largest when the latter is strong. In this region, the performance of discriminant saliency (.85) is close to 90% of that of humans (.95), while the other two detectors only achieve close to 85% (.81).

## 4.2   Discriminant saliency on motion fields

Similarly to the static case, center-surround discriminant saliency can produce motion-based saliency maps if combined with motion features. We have implemented a simple motion-based detector by computing a dense motion vector map (optical flow) between pairs of consecutive images, and using the magnitude of the motion vector at each location as motion feature. The probability distributions of this feature, within center and surround, were estimated with histograms, and the motion saliency maps computed with (2).

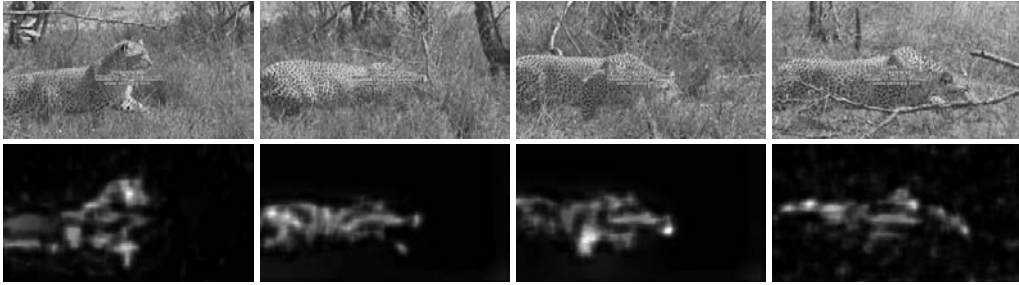

Figure 5: Optical flow-based saliency in the presence of egomotion.

Despite the simplicity of our motion representation, the discriminant saliency detector exhibits interesting performance. Figure 5 shows several frames (top row) from a video sequence, and their discriminant motion saliency maps (bottom row). The sequence depicts a leopard running in a grassland, which is tracked by a moving camera. This results in significant variability of the background, due to egomotion, making the detection of foreground motion (leopard), a non-trivial task. As shown in the saliency maps, discriminant saliency successfully disregards the egomotion component of the optical flow, detecting the leopard as most salient.

### 4.3   Discriminant Saliency with dynamic background

While the results of Figure 5 are probably within the reach of previously proposed saliency models, they illustrate the flexibility of discriminant saliency. In this section we move to a domain where traditional saliency algorithms almost invariably fail. This consists of videos of scenes with complex and dynamic backgrounds (e.g. water waves, or tree leaves). In order to capture the motion patterns characteristic of these backgrounds it is necessary to rely on reasonably sophisticated probabilistic models, such as the dynamic texture model [5]. Such models are very difficult to fit in the conventional, e.g. difference-based, saliency frameworks but naturally compatible with the discriminant saliency hypothesis. We next combine discriminant center-surround saliency with the dynamic texture model, to produce a background-subtraction algorithm for scenes with complex background dynamics. While background subtraction is a classic problem in computer vision, there has been relatively little progress for these type of scenes (e.g. see [15] for a review).

A dynamic texture (DT) [5, 3] is an autoregressive, generative model for video. It models the spatial component of the video and the underlying temporal dynamics as two stochastic processes. A video is represented as a time-evolving state process $x_t \in \mathbb{R}^n$, and the appearance of a frame $y_t \in \mathbb{R}^m$ is a linear function of the current state vector with some observation noise. The system equations are

$$
\begin{aligned}
x_t &= Ax_{t-1} + v_t \\
y_t &= Cx_t + w_t
\end{aligned}
\tag{3}
$$

where $A \in \mathbb{R}^{n \times n}$ is the state transition matrix, $C \in \mathbb{R}^{m \times n}$ is the observation matrix. The state and observation noise are given by $v_t \sim_{iid} \mathcal{N}(0, Q,)$ and $w_t \sim_{iid} \mathcal{N}(0, R)$, respectively. Finally, the initial condition is distributed as $x_1 \sim \mathcal{N}(\mu, S)$. Given a sequence of images, the parameters of the dynamic texture can be learned for the center and surround regions at each image location, enabling a probabilistic description of the video, with which the mutual information of (2) can be evaluated.

We applied the dynamic texture-based discriminant saliency (DTDS) detector to three video sequences containing objects moving in water. The first (Water-Bottle from [23]) depicts a bottle floating in water which is hit by rain drops, as shown in Figure 7(a). The second and third, Boat and Surfer, are composed of boats/surfers moving in water, and shown in Figure 8(a) and 9(a). These sequences are more challenging, since the micro-texture of the water surface is superimposed on a lower frequency sweeping wave (Surfer) and interspersed with high frequency components due to turbulent wakes (created by the boat, surfer, and crest of the sweeping wave). Figures 7(b), 8(b) and 9(b), show the saliency maps produced by discriminant saliency for the three sequences. The DTDS detector performs surprisingly well, in all cases, at detecting the foreground objects while ignoring the movements of the background. In fact, the DTDS detector is close to an ideal background-subtraction algorithm for these scenes.

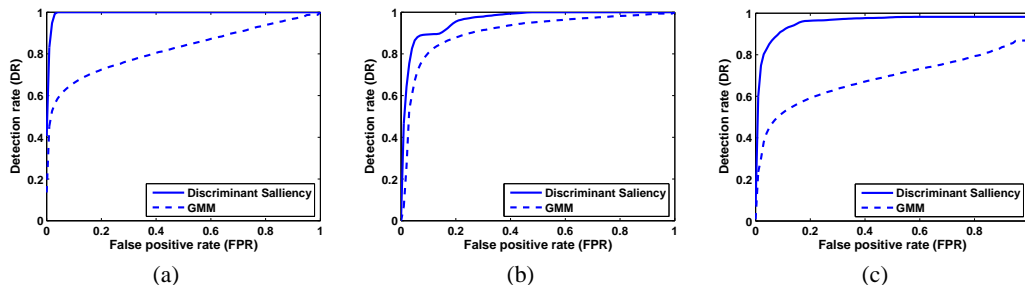

Figure 6: Performance of background subtraction algorithms on: (a) Water-Bottle, (b) Boat, and (c) Surfer.

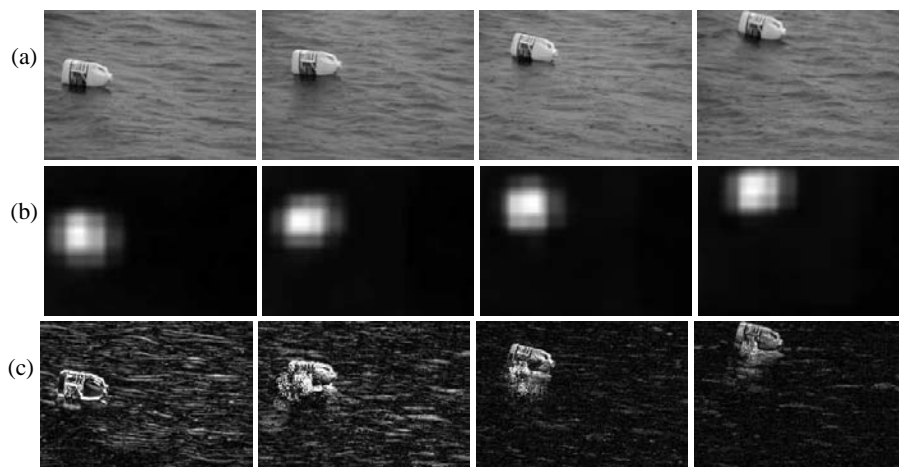

Figure 7: Results on Bottle: (a) original; b) discriminant saliency with DT; and c) GMM model of [16, 24].

For comparison, we present the output of a state-of-the-art background subtraction algorithm, a Gaussian mixture model (GMM) [16, 24]. As can be seen in Figures 7(c), 8(c) and 9(c), the resulting foreground detection is very noisy, and cannot adapt to the highly dynamic nature of the water surface. Note, in particular, that the waves produced by boat and surfer, as well as the sweeping wave crest, create serious difficulties for this algorithm. Unlike the saliency maps of DTDS, the resulting foreground maps would be difficult to analyze by subsequent vision (e.g. object tracking) modules. To produce a quantitative comparison of the saliency maps, these were thresholded at a large range of values. The results were compared with ground-truth foreground masks, and an ROC curve produced for each algorithm. The results are shown in Figure 6, where it is clear that while DTDS tends to do well on these videos, the GMM based background model does fairly poorly.

## Footnotes

[1]Note that this approximation *does not* assume that the features are independently distributed, but simply that their dependencies are not informative about the class.

[2]For the computation of the discriminant saliency maps, we followed the common practice of psychophysics and physiology [18, 10], to set the size of the center window to a value *comparable* to that of the display items, and the size of the surround window is 6 times of that of the center. Informal experimentation has shown that the saliency results are not substantively affected by variations around the parameter values adopted.

[3]Results obtained with the MATLAB implementation available in [22].

# References

[1] N. D. Bruce and J. K. Tsotsos. Saliency based on information maximization. In *Proc. NIPS*, 2005.

[2] R. Buccigrossi and E. Simoncelli. Image compression via joint statistical characterization in the wavelet domain. *IEEE Transactions on Image Processing*, 8:1688–1701, 1999.

[3] A. B. Chan and N. Vasconcelos. Modeling, clustering, and segmenting video with mixtures of dynamic textures. *IEEE Trans. PAMI*, In Press.

[4] M. N. Do and M. Vetterli. Wavelet-based texture retrieval using generalized gaussian density and kullback-leibler distance. *IEEE Trans. Image Processing*, 11(2):146–158, 2002.

[5] G. Doretto, A. Chiuso, Y. N. Wu, and S. Soatto. Dynamic textures. *Int. J. Comput. Vis.*, 51, 2003.

[6] D. Gao and N. Vasconcelos. Discriminant saliency for visual recognition from cluttered scenes. In *Proc. NIPS*, pages 481–488, 2004.

[7] D. Gao and N. Vasconcelos. Decision-theoretic saliency: computational principle, biological plausibility, and implications for neurophysiology and psychophysics. submitted to *Neural Computation*, 2007.

[8] J. Harel, C. Koch, and P. Perona. Graph-based visual saliency. In *Proc. NIPS*, 2006.

[9] J. Huang and D. Mumford. Statistics of Natural Images and Models. In *Proc. IEEE Conf. CVPR*, 1999.

[10] D. H. Hubel and T. N. Wiesel. Receptive fields and functional architecture in two nonstriate visual areas (18 and 19) of the cat. *J. Neurophysiol.*, 28:229–289, 1965.

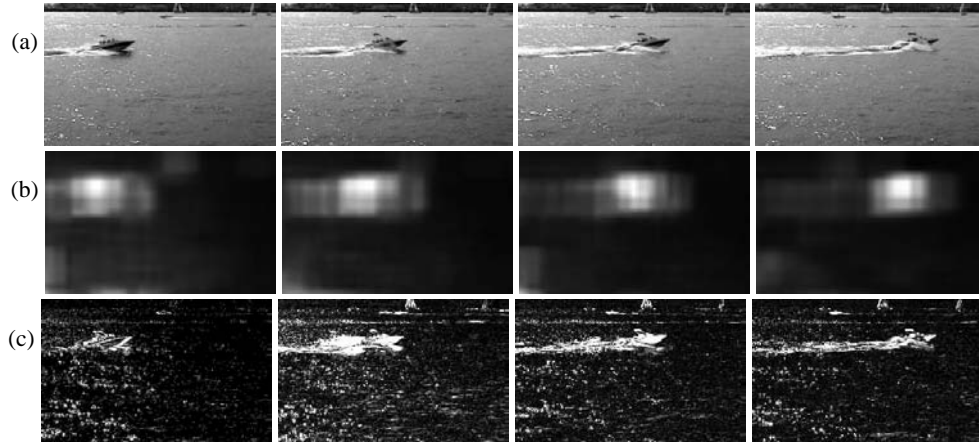

Figure 8: Results on Boats: (a) original; b) discriminant saliency with DT; and c) GMM model of [16, 24].

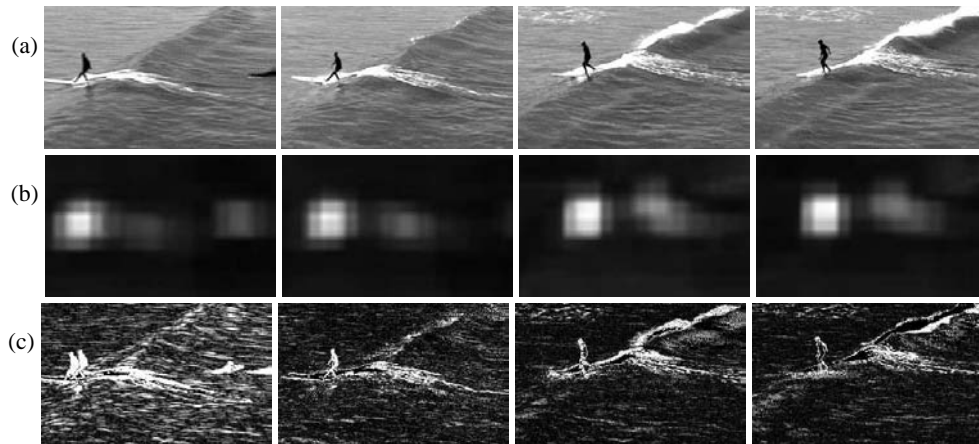

Figure 9: Results on Surfer: (a) original; b) discriminant saliency with DT; and c) GMM model of [16, 24].

[11] L. Itti and C. Koch. A saliency-based search mechanism for overt and covert shifts of visual attention. *Vision Research*, 40:1489–1506, 2000.

[12] L. Itti, C. Koch, and E. Niebur. A model of saliency-based visual attention for rapid scene analysis. *IEEE Trans. PAMI*, 20(11), 1998.

[13] S. G. Mallat. A theory for multiresolution signal decomposition: The wavelet representation. *IEEE Trans. PAMI*, 11(7):674–693, 1989.

[14] H. C. Nothdurft. The conspicuousness of orientation and motion contrast. *Spat. Vis.*, 7, 1993.

[15] Y. Sheikh and M. Shah. Bayesian modeling of dynamic scenes for object detection. *IEEE Trans. on PAMI*, 27(11):1778–92, 2005.

[16] C. Stauffer and W. Grimson. Adaptive background mixture models for real-time tracking. In *CVPR*, pages 246–52, 1999.

[17] B. W. Tatler, R. J. Baddeley, and I. D. Gilchrist. Visual correlates of fixation selection: effects of scale and time. *Vision Research*, 45:643–659, 2005.

[18] A. Treisman and G. Gelade. A feature-integratrion theory of attention. *Cognit. Psych.*, 12, 1980.

[19] A. Treisman and S. Gormican. Feature analysis in early vision: Evidence from search asymmetries. *Psychological Review*, 95:14–58, 1988.

[20] A. Tversky. Features of similarity. *Psychol. Rev.*, 84, 1977.

[21] N. Vasconcelos. Scalable discriminant feature selection for image retrieval. In *CVPR*, 2004.

[22] D. Walther and C. Koch. Modeling attention to salient proto-objects. *Neural Networks*, 19, 2006.

[23] J. Zhong and S. Sclaroff. Segmenting foreground objects from a dynamic textured background via a robust Kalman filter. In *ICCV*, 2003.

[24] Z. Zivkovic. Improved adaptive Gaussian mixture model for background subtraction. In *ICVR*, 2004.

